# Learning and using relational theories

**Charles Kemp, Noah D. Goodman & Joshua B. Tenenbaum**
Department of Brain and Cognitive Sciences, MIT, Cambridge, MA 02139
`{ckemp,ndg,jbt}@mit.edu`

## Abstract

Much of human knowledge is organized into sophisticated systems that are often called intuitive theories. We propose that intuitive theories are mentally represented in a logical language, and that the subjective complexity of a theory is determined by the length of its representation in this language. This complexity measure helps to explain how theories are learned from relational data, and how they support inductive inferences about unobserved relations. We describe two experiments that test our approach, and show that it provides a better account of human learning and reasoning than an approach developed by Goodman [1].

What is a theory, and what makes one theory better than another? Questions like these are of obvious interest to philosophers of science but are also discussed by psychologists, who have argued that everyday knowledge is organized into rich and complex systems that are similar in many respects to scientific theories. Even young children, for instance, have systematic beliefs about domains including folk physics, folk biology, and folk psychology [2]. Intuitive theories like these play many of the same roles as scientific theories: in particular, both kinds of theories are used to explain and encode observations of the world, and to predict future observations.

This paper explores the nature, use and acquisition of simple theories. Consider, for instance, an anthropologist who has just begun to study the social structure of a remote tribe, and observes that certain words are used to indicate relationships between selected pairs of individuals. Suppose that term $T1(\cdot, \cdot)$ can be glossed as $\texttt{ancestor}(\cdot, \cdot)$, and that $T2(\cdot, \cdot)$ can be glossed as $\texttt{friend}(\cdot, \cdot)$. The anthropologist might discover that the first term is transitive, and that the second term is symmetric with a few exceptions. Suppose that term $T3(\cdot, \cdot)$ can be glossed as $\texttt{defers\_to}(\cdot, \cdot)$, and that the tribe divides into two castes such that members of the second caste defer to members of the first caste. In this case the anthropologist might discover two latent concepts ($\texttt{caste\_1}(\cdot)$ and $\texttt{caste\_2}(\cdot)$) along with the relationship between these concepts.

As these examples suggest, a theory can be defined as a system of laws and concepts that specify the relationships between the elements in some domain [2]. We will consider how these theories are learned, how they are used to encode relational data, and how they support predictions about unobserved relations. Our approach to all three problems relies on the notion of subjective complexity. We propose that theory learners prefer simple theories, that people remember relational data in terms of the simplest underlying theory, and that people extend a partially observed data set according to the simplest theory that is consistent with their observations. There is no guarantee that a single measure of subjective complexity can do all of the work that we require [3]. This paper, however, explores the strong hypothesis that a single measure will suffice.

Our formal treatment of subjective complexity begins with the question of how theories are mentally represented. We suggest that theories are represented in some logical language, and propose a specific first-order language that serves as a hypothesis about the "language of thought." We then pursue the idea that the subjective complexity of a theory corresponds to the length of its representation in this language. Our approach therefore builds on the work of Feldman [4], and is related to other psychological applications of the notion of Kolmogorov complexity [5]. The complexity measure we describe can be used to define a probability distribution over a space of theories, and we develop a model of theory acquisition by using this distribution as the prior for a Bayesian learner. We also

**(a) Star**

```
   22 33 44 55 66 77 88
11
   21 31 41 51 61 71 81
```

$R(X, X)$.
$R(X, 1)$.

**(b) Bipartite**

```
16 26 36 46 56
17 27 37 47 57
18 28 38 48 58
```

$T(6)$. $T(7)$. $T(8)$.
$R(X, Y) \leftarrow \bar{T}(X), T(Y)$.

**(c) Exception**

```
11   26 36 46 56
   17 27 37 47 57
   18 28 38 48 58
```

$T(6)$. $T(7)$. $T(8)$.
$R(X, Y) \leftarrow \bar{T}(X), T(Y)$.
$R(1, 1)$. $\bar{R}(1, 6)$.

**(d) Symmetric**

```
11 22 33 44 55 66 77
                13 31
 12 21
        24 42
                56 65
```

$R(1, 2)$. $R(1, 3)$. $R(2, 4)$. $R(5, 6)$.
$R(X, Y) \leftarrow R(Y, X)$.
$R(X, X)$.

**(e) Transitive**

```
   12
   13 23
   14 24 34
   15 25 35 45
   16 26 36 46 56
```

$R(1, 2)$. $R(2, 3)$. $R(3, 4)$.
$R(4, 5)$. $R(5, 6)$.
$R(X, Z) \leftarrow R(X, Y), R(Y, Z)$.

**(f) Random**

```
   21
   13 32
   14 24 34
   51 52 35 54
   61 26 63 46 56
```

$R(5, X)$. $R(X, 4)$.
$R(2, 1)$. $R(1, 3)$. $R(6, 1)$. $R(3, 2)$.
$R(2, 6)$. $R(3, 5)$. $R(6, 3)$. $R(4, 6)$.
$\bar{R}(X, X)$. $\bar{R}(6, 4)$. $\bar{R}(5, 3)$.

Figure 1: Six possible extensions for a binary predicate $R(\cdot, \cdot)$. In each case, the objects in the domain are represented as digits, and a pair such as 16 indicates that $R(1, 6)$ is true. Below each set of pairs, the simplest theory according to our complexity measure is shown.

show how the same Bayesian approach helps to explain how theories support inductive generalization: given a set of observations, future observations (e.g. whether one individual defers to another) can be predicted using the posterior distribution over the space of theories.

We test our approach by developing two experiments where people learn and make predictions about binary and ternary relations. As far as we know, the approach of Goodman [1] is the only other measure of theory complexity that has previously been tested as a psychological model [6]. We show that our experiments support our approach and raise challenges for this alternative model.

## 1 Theory complexity: a representation length approach

Intuitive theories correspond to mental representations of some sort, and our first task is to characterize the elements used to build these representations. We explore the idea that a theory is a system of statements in a logical language, and six examples are shown in Fig. 1. The theory in Fig. 1b is related to the defers_to$(\cdot, \cdot)$ example already described. Here we are interested in a domain including 9 elements, and a two place predicate $R(\cdot, \cdot)$ that is true of all and only the 15 pairs shown. R is defined using a unary predicate T which is true of only three elements: 6, 7, and 8. The theory includes a clause which states that $R(X, Y)$ is true for all pairs XY such that $T(X)$ is false and $T(Y)$ is true. The theory in Fig. 1c is very similar, but includes an additional clause which specifies that $R(1, 1)$ is true, and an exception which specifies that $R(1, 6)$ is false. Formally, each theory we consider is a collection of function-free definite clauses. All variables are universally quantified: for instance, the clause $R(X, Z) \leftarrow R(X, Y), R(Y, Z)$ is equivalent to the logical formula $\forall x \, \forall y \, \forall z \, (R(x, z) \leftarrow R(x, y) \land R(y, z))$. For readability, the theories in Fig. 1 include parentheses and arrows, but note that these symbols are unnecessary and can be removed. Our proposed language includes only predicate symbols, variable symbols, constant symbols, and a period that indicates when one clause finishes and another begins.

Each theory in Fig. 1 specifies the extension of one or more predicates. The extension of predicate P is defined in terms of predicate $P^+$ (which captures the basic rules that lead to membership in P) and predicate $P^-$ (which captures exceptions to these rules). The resulting extension of P is defined

as $P^+ \setminus P^-$, or the set difference of $P^+$ and $P^-$.[1] Once P has been defined, later clauses in the theory may refer to P or its negation $\bar{P}$. To ensure that our semantics is well-defined, the predicates in any valid theory must permit an ordering so that the definition of any predicate does not refer to predicates that follow it in the order. Formally, the definition of each predicate $P^+$ or $P^-$ can refer only to itself (recursive definitions are allowed) and to any predicate M or $\bar{M}$ where $M < P$.

Once we have committed to a specific language, the subjective complexity of a theory is assumed to correspond to the number of symbols in its representation. We have chosen a language where there is one symbol for each position in a theory where a predicate, variable or constant appears, and one symbol to indicate when each clause ends. Given this language, the subjective complexity $c(T)$ of theory $T$ is equal to the sum of the number of clauses in the theory and the number of positions in the theory where a predicate, variable or constant appears:

$$c(T) = \texttt{\#clauses}(T) + \texttt{\#pred\_slots}(T) + \texttt{\#var\_slots}(T) + \texttt{\#const\_slots}(T). \quad (1)$$

For instance, the clause $R(X, Z) \leftarrow R(X, Y), R(Y, Z).$ contributes ten symbols towards the complexity of a theory (three predicate symbols, six variable symbols, and one period). Other languages might be considered: for instance, we could use a language which uses five symbols (e.g. five bits) to represent each predicate, variable and constant, and one symbol (e.g. one bit) to indicate the end of a clause. Our approach to subjective complexity depends critically on the representation language, but once a language has been chosen the complexity measure is uniquely specified.

Although our approach is closely related to the notion of Kolmogorov complexity and to Minimum Message Length (MML) and Minimum Description Length (MDL) approaches, we refer to it as a Representation Length (RL) approach. A RL approach includes a commitment to a specific language that is proposed as a psychological hypothesis, but these other approaches aspire towards results that do not depend on the language chosen.[2] It is sometimes suggested that the notion of Kolmogorov complexity provides a more suitable framework for psychological research than the RL approach, precisely because it allows for results that do not depend on a specific description language [8]. We subscribe to the opposite view. Mental representations presumably rely on some particular language, and identifying this language is a central challenge for psychological research.

The language we described should be considered as a tentative approximation of the language of thought. Other languages can and should be explored, but our language has several appealing properties. Feldman [4] has argued that definite clauses are psychologically natural, and working with these representations allows our approach to account for several classic results from the concept learning literature. For instance, our language leads to the prediction that conjunctive concepts are easier to learn than disjunctive concepts [9].[3] Working with definite clauses also ensures that each of our theories has a unique minimal model, which means that the extension of a theory can be defined in a particularly simple way. Finally, human learners deal gracefully with noise and exceptions, and our language provides a simple way to handle exceptions.

Any concrete proposal about the language of thought should make predictions about memory, learning and reasoning. Suppose that data set $D$ lists the extensions of one or more predicates, and that a theory is a "candidate theory" for $D$ if it correctly defines the extensions of all predicates in $D$. Note that a candidate theory may well include latent predicates—predicates that do not appear in $D$, but are useful for defining the predicates that have been observed. We will assume that humans encode $D$ in terms of the simplest candidate theory for $D$, and that the difficulty of memorizing $D$ is determined by the subjective complexity of this theory. Our approach can and should be tested against classic results from the memory literature. Unlike some other approaches to complexity [10], for instance, our model predicts that a sequence of $k$ items is about equally easy to remember regardless of whether the items are drawn from a set of size 2, a set of size 10, or a set of size 1000 [11].

To develop a model of inductive learning and reasoning, we take a Bayesian approach, and use our complexity measure to define a prior distribution over a hypothesis space of theories: $P(T) \propto 2^{-c(T)}$.[4] Given this prior distribution, we can use Bayesian inference to make predictions about unobserved relations and to discover the theory $T$ that best accounts for the observations in data set $D$ [12, 13]. Suppose that we have a likelihood function $P(D|T)$ which specifies how the examples in $D$ were generated from some underlying theory $T$. The best explanation for the data $D$ is the theory that maximizes the posterior distribution $P(T|D) \propto P(D|T)P(T)$. If we need to predict whether ground term $g$ is likely to be true,[5] we can sum over the space of theories:

$$P(g|D) = \sum_{T} P(g|T)P(T|D) = \frac{1}{P(D)} \sum_{T:g \in T} P(D|T)P(T) \tag{2}$$

where the final sum is over all theories $T$ that make ground term $g$ true.

## 1.1 Related work

The theories we consider are closely related to logic programs, and methods for Inductive Logic Programming (ILP) explore how these programs can be learned from examples [14]. ILP algorithms are often inspired by the idea of searching for the shortest theory that accounts for the available data, and ILP is occasionally cast as the problem of minimizing an explicit MDL criterion [10]. Although ILP algorithms are rarely considered as cognitive models, the RL approach has a long psychological history, and is proposed by Chomsky [15] and Leeuwenberg [16] among others.

Formal measures of complexity have been developed in many fields [17], and there is at least one other psychological account of theory complexity. Goodman [1] developed a complexity measure that was originally a philosophical proposal about scientific theories, but was later tested as a model of subjective complexity [6]. A detailed description of this measure is not possible here, but we attempt to give a flavor of the approach. Suppose that a *basis* is a set of predicates. The starting point for Goodman's model is the intuition that basis $B1$ is at least as complex as basis $B2$ if $B1$ can be used to define $B2$. Goodman argues that this intuition is flawed, but his model is founded on a refinement of this intuition. For instance, since the binary predicate in Fig. 1b can be defined in terms of two unary predicates, Goodman's approach requires that the complexity of the binary predicate is no more than the sum of the complexities of the two unary predicates.

We will use Goodman's model as a baseline for evaluating our own approach, and a comparison between these two models should be informed by both theoretical and empirical considerations. On the theoretical side, our approach relies on a simple principle for deciding which structural properties are relevant to the measurement of complexity: the relevant properties are those with short logical representations. Goodman's approach incorporates no such principle, and he proposes somewhat arbitrarily that reflexivity and symmetry are among the relevant structural properties but that transitivity is not. A second reason for preferring our model is that it makes contact with a general principle—the idea that simplicity is related to representation length—that has found many applications across psychology, machine learning, and philosophy.

## 2 Experimental results

We designed two experiments to explore settings where people learn, remember, and make inductive inferences about relational data. Although theories often consist of systems of many interlocking relations, we keep our experiments simple by asking subjects to learn and reason about a single relation at a time. Despite this restriction, our experiments still make contact with several issues raised by systems of relations. As the defers_to$(\cdot, \cdot)$ example suggests, a single relation may be best explained as the observable tip of a system involving several latent predicates (e.g. caste_1$(\cdot)$ and caste_2$(\cdot)$).

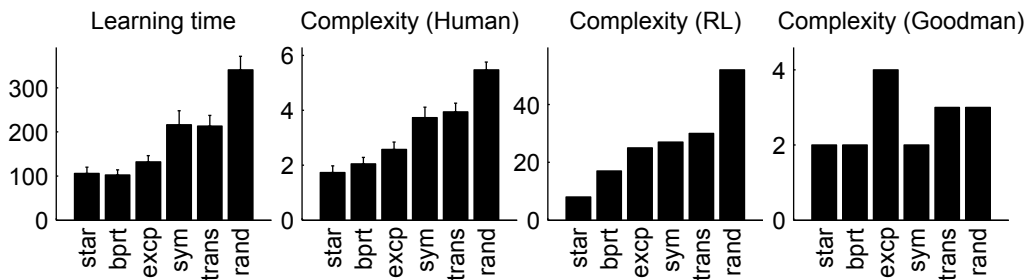

Figure 2: (a) Average time in seconds to learn the six sets in Fig. 1. (b) Average ratings of set complexity. (c) Complexity scores according to our representation length (RL) model. (d) Complexity scores according to Goodman's model.

## 2.1 Experiment 1: memory and induction

In our first experiment, we studied the subjective complexity of six binary relations that display a range of structural properties, including reflexivity, symmetry, and transitivity.

**Materials and Methods.** 18 adults participated in this experiment. Subjects were required to learn the 6 sets shown in Fig. 1, and to make inductive inferences about each set. Although Fig. 1 shows pairs of digits, the experiment used letter pairs, and the letters for each condition and the order in which these conditions were presented were randomized across subjects. The pairs for each condition were initially laid out randomly on screen, and subjects could drag them around and organize them to help them understand the structure of the set. At any stage, subjects could enter a test phase where they were asked to list the 15 pairs belonging to the current set. Subjects who made an error on the test were returned to the learning phase. After 9 minutes had elapsed, subjects were allowed to pass the test regardless of how many errors they made.

After passing the test, subjects were asked to rate the complexity of the set compared to other sets with 15 pairs. Ratings were provided on a 7 point scale. Subjects were then asked to imagine that a new letter (e.g. letter 9) had belonged to the current alphabet, and were given two inductive tasks. First they were asked to enter between 1 and 10 novel pairs that they might have expected to see (each novel pair was required to include the new letter). Next they were told about a novel pair that belonged to the set (e.g. pair 91), and were again asked to enter up to 10 additional pairs that they might have expected to see.

**Results.** The average time needed to learn each set is shown in Fig. 2a, and ratings of set complexity are shown in Fig. 2b. It is encouraging that these measures yield converging results, but they may be confounded since subjects rated the complexity of a set immediately after learning it. The complexities plotted in Fig. 2c are the complexities of the theories shown in Fig. 1, which we believe to be the simplest theories according to our complexity measure. The final plot in Fig. 2 shows complexities according to Goodman's model, which assigns each binary relation an integer between 0 and 4. There are several differences between these models: for instance, Goodman's account incorrectly predicts that the exception case is the hardest of the six, but our model acknowledges that a simple theory remains simple if a handful of exceptions are added. Goodman's account also predicts that transitivity is not an important structural regularity, but our model correctly predicts that the transitive set is simpler than the same set with some of the pairs reversed (the random set).

Results for the inductive task are shown in Fig. 3. The first two columns show the number of subjects who listed each novel pair. The remaining two columns show the probability of set membership predicted by our model. To generate these predictions, we applied Equation 2 and summed over a set of theories created by systematically extending the theories shown in Fig. 1. Each extended theory includes up to one additional clause for each predicate in the base theory, and each additional clause includes at most two predicate slots. For instance, each extended theory for the bipartite case is created by choosing whether or not to add the clause $\texttt{T(9)}$, and adding up to one clause for predicate $\texttt{R}$.[6] For the first inductive task, the likelihood term $P(D|T)$ (see Equation 2) is set to 0 for all theories that are not consistent with the pairs observed during training, and to a constant for all remaining theories. For the second task we assumed in addition that the novel pair observed is

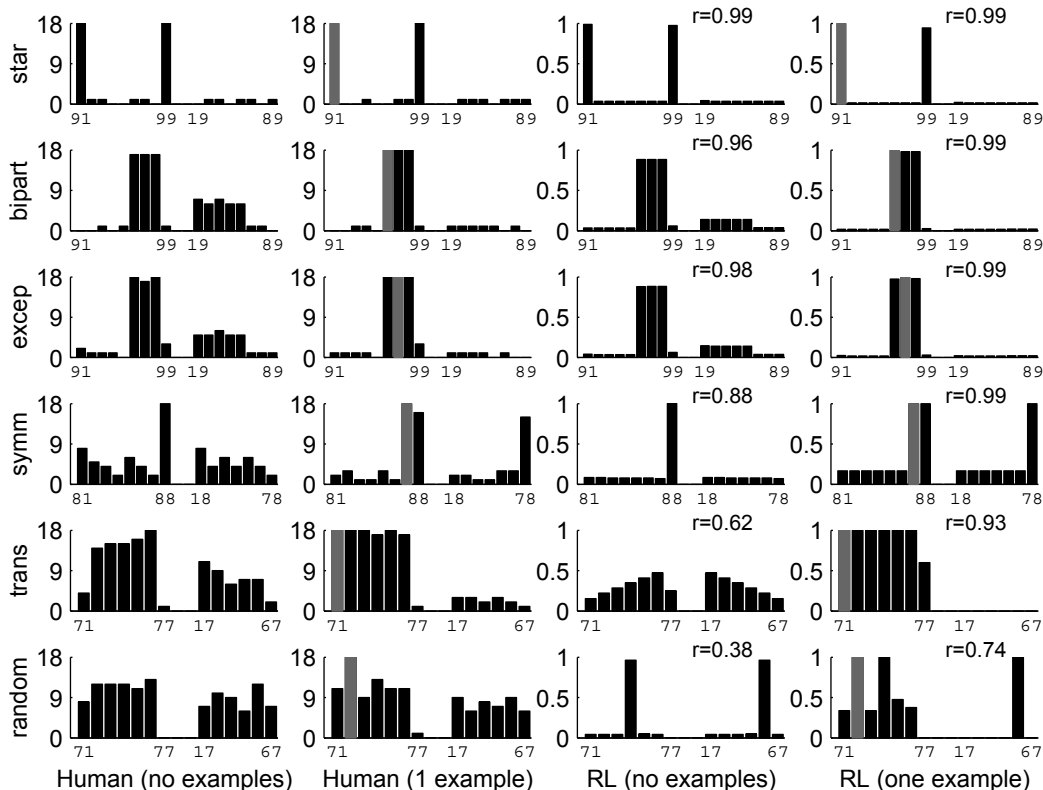

Figure 3: Data and model predictions for the induction task in Experiment 1. Columns 1 and 3 show predictions before any pairs involving the new letter are observed. Columns 2 and 4 show predictions after a single novel pair (marked with a gray bar) is observed to belong to the set. The model plots for each condition include correlations with the human data.

sampled at random from all pairs involving the new letter.[7] All model predictions were computed using Mace4 [18] to generate the extension of each theory considered.

The supporting material includes predictions for a model based on the Goodman complexity measure and an exemplar model which assumes that the new letter will be just like one of the old letters.[8] The exemplar model outperforms our model in the random condition, and makes accurate predictions about three other conditions. Overall, however, our model performs better than the two baselines. Here we focus on two important predictions that are not well handled by the exemplar model. In the symmetry condition, almost all subjects predict that 78 belongs to the set after learning that 87 belongs to the set, suggesting that they have learned an abstract rule. In the transitive condition, most subjects predict that pairs 72 through 76 belong to the set after learning that 71 belongs to the set. Our model accounts for this result, but the exemplar model has no basis for making predictions about letter 7, since this letter is now known to be unlike any of the others.

## 2.2 Experiment 2: learning from positive examples

During the learning phase of our first experiment, subjects learned a theory based on positive examples (the theory included all pairs they had seen) and negative examples (the theory ruled out all pairs they had not seen). Often, however, humans learn theories based on positive examples alone. Suppose, for instance, that our anthropologist has spent only a few hours with a new tribe. She may have observed several pairs who are obviously friends, but should realize that many other pairs of friends have not yet interacted in her presence.

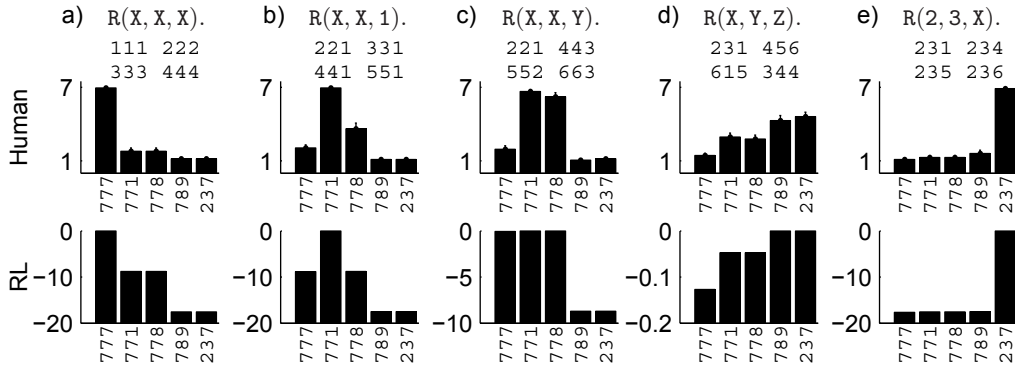

Figure 4: Data and model predictions for Experiment 2. The four triples observed for each set are shown at the top of the figure. The first row of plots shows average ratings on a scale from 1 (very unlikely to belong to the set) to 7 (very likely). Model predictions are plotted as log probabilities.

Our framework can handle cases like these if we assume that the data $D$ in Equation 2 are sampled from the ground terms that are true according to the underlying theory. We follow [10] and [13] and use a distribution $P(D|T)$ which assumes that the examples in $D$ are randomly sampled with replacement from the ground terms that are true. This sampling assumption encourages our model to identify the theory with the smallest extension that is compatible with all of the training examples. We tested this approach by designing an experiment where learners were given sets of examples that were compatible with several underlying theories.

**Materials and Methods.** 15 adults participated in this experiment immediately after taking Experiment 1. In each of five conditions, subjects were told about a set of triples built from an alphabet of 9 letters. They were shown four triples that belonged to the set (Fig. 4), and told that the set might include triples that they had not seen. Subjects then gave ratings on a seven point scale to indicate whether five additional triples (see Fig. 4) were likely to belong to the set.

**Results.** Average ratings and model predictions are shown in Fig. 4. Model predictions for each condition were computed using Equation 2 and summing over a space of theories that included the five theories shown at the top of Fig. 4, variants of these five theories which stated that certain pairs of slots could not be occupied by the same constant,[9] and theories that included no variables but merely enumerated up to 5 triples.[10]

Although there are general theories like R(X, Y, Z) that are compatible with the triples observed in all five conditions, Fig. 4 shows that people were sensitive to different regularities in each case.[11] We focus on one condition (Fig. 4b) that exposes the strengths and weaknesses of our model. According to our model, the two most probable theories given the triples for this condition are R(X, X, 1) and the closely related variant that rules out R(1, 1, 1). The next most probable theory is R(X, X, Y). These predictions are consistent with people's judgments that 771 is very likely to belong to the set, and that 778 is the next most likely option. Unlike our model, however, people consider 777 to be substantially less likely than 778 to belong to the set. This result may suggest that the variant of R(X, X, Y) that rules out R(X, X, X) deserves a higher prior probability than our model recognizes. To better account for cases like this, it may be worth considering languages where any two variables that belong to the same clause but have different names must refer to different entities.

## 3 Discussion and Conclusion

There are many psychological models of concept learning [4, 12, 13], but few that use representations rich enough to capture the content of intuitive theories. We suggested that intuitive theories are mentally represented in a first-order logical language, and proposed a specific hypothesis about

this "language of thought." We assumed that the subjective complexity of a theory depends on the length of its representation in this language, and described experiments which suggest that the resulting complexity measure helps to explain how theories are learned and used for inductive inference.

Our experiments deliberately used stimuli that minimize the influence of prior knowledge. Theories, however, are cumulative, and the theory that seems simplest to a learner will often depend on her background knowledge. Our approach provides a natural place for background knowledge to be inserted. A learner can be supplied with a stock of background predicates, and the shortest representation for a data set will depend on which background predicates are available. Since different sets of predicates will lead to different predictions about subjective complexity, empirical results can help to determine the background knowledge that people bring to a given class of problems.

Future work should aim to refine the representation language and complexity measure we proposed. We expect that something like our approach will be suitable for modeling a broad class of intuitive theories, but the specific framework presented here can almost certainly be improved. Future work should also consider different strategies for searching the space of theories. Some of the strategies developed in the ILP literature should be relevant [14], but a detailed investigation of search algorithms seems premature until our approach has held up to additional empirical tests. It is comparatively easy to establish whether the theories that are simple according to our approach are also considered simple by people, and our experiments have made a start in this direction. It is much harder to establish that our approach captures most of the theories that are subjectively simple, and more exhaustive experiments are needed before this conclusion can be drawn.

Boolean concept learning has been studied for more than fifty years [4, 9], and many psychologists have made empirical and theoretical contributions to this field. An even greater effort will be needed to crack the problem of theory learning, since the space of intuitive theories is much richer than the space of Boolean concepts. The difficulty of this problem should not be underestimated, but computational approaches can contribute part of the solution.

**Acknowledgments** Supported by the William Asbjornsen Albert memorial fellowship (CK), the James S. Mc-Donnell Foundation Causal Learning Collaborative Initiative (NDG, JBT) and the Paul E. Newton chair (JBT).

## Footnotes

[1] The extension of $P^+$ is the smallest set that satisfies all of the clauses that define $P^+$, and the extension of $P^-$ is defined similarly. To simplify our notation, Fig. 1 uses P to refer to both P and $P^+$, and $\bar{P}$ to refer to $\bar{P}$ and $P^-$. Any instance of P that appears in a clause defining P is really an instance of $P^+$, and any instance of $\bar{P}$ that appears in a clause defining $\bar{P}$ is really an instance of $P^-$.

[2] MDL approaches also commit to a specific language, but this language is often intended to be as general as possible. See, for instance, the discussion of universal codes in Grünwald et al. [7].

[3] A conjunctive concept $C(\cdot)$ can be defined using a single clause: $C(X) \leftarrow A(X), B(X)$. The shortest definition of a disjunctive concept requires two clauses: $D(X) \leftarrow A(X). D(X) \leftarrow B(X)$.

[4]To ensure that this distribution can be normalized, we assume that there is some upper bound on the number of predicate symbols, variable symbols, and constants, and on the length of the theories we will consider. There will therefore be a finite number of possible theories, and our prior will be a valid probability distribution.

[5]A ground term is a term such as R(8, 9) that does not include any variables.

[6]$\texttt{R(9, X)}$, $\bar{\texttt{R}}\texttt{(2, 9)}$, and $\texttt{R(X, 9)} \leftarrow \texttt{R(X, 2)}$ are three possible additions.

[7]For the second task, $P(D|T)$ is set to 0 for theories that are inconsistent with the training pairs and theories which do not include the observed novel pair. For all remaining theories, $P(D|T)$ is set to $\frac{1}{n}$, where $n$ is the total number of novel pairs that are consistent with $T$.

[8]Supporting material is available at www.charleskemp.com

[9] One such theory includes two clauses: R(X, X, Y). R̄(X, X, X).

[10] One such theory is the following list of clauses: R(2, 2, 1). R(3, 3, 1). R(4, 4, 1). R(5, 5, 1). R(7, 7, 7).

[11] Similar results have been found with 9-month old infants. Cases like Figs. 4b and 4c have been tested in an infant language-learning study where the stimuli were three-syllable strings [19]. 9-month old infants exposed to strings like the four in Fig. 4c generalized to other strings consistent with the theory R(X, X, Y), but infants in the condition corresponding to Fig. 4b generalized only to strings consistent with the theory R(X, X, 1).

## References

[1] N. Goodman. *The structure of appearance*. 2nd edition, 1961.

[2] S. Carey. *Conceptual change in childhood*. MIT Press, Cambridge, MA, 1985.

[3] H. A. Simon. Complexity and the representation of patterned sequences of symbols. *Psychological Review*, 79:369–382, 1972.

[4] J. Feldman. An algebra of human concept learning. *JMP*, 50:339–368, 2006.

[5] N. Chater and P. Vitanyi. Simplicity: a unifying principle in cognitive science. *TICS*, 7:19–22, 2003.

[6] J. T. Krueger. *A theory of structural simplicity and its relevance to aspects of memory, perception, and conceptual naturalness*. PhD thesis, University of Pennsylvania, 1979.

[7] P. Grünwald, I. J. Myung, and M. Pitt, editors. *Advances in Minimum Description Length: Theory and Applications*. 2005.

[8] N. Chater. Reconciling simplicity and likelihood principles in perceptual organization. *Psychological Review*, 103:566–581, 1996.

[9] J. A. Bruner, J. S. Goodnow, and G. J. Austin. *A study of thinking*. Wiley, 1956.

[10] D. Conklin and I. H. Witten. Complexity-based induction. *Machine Learning*, 16(3):203–225, 1994.

[11] G. A. Miller. The magical number seven, plus or minus two: Some limits on our capacity for processing information. *Psychological Review*, 63(1):81–97, 1956.

[12] N. D. Goodman, T. L. Griffiths, J. Feldman, and J. B. Tenenbaum. A rational analysis of rule-based concept learning. In *CogSci*, 2007.

[13] J. B. Tenenbaum and T. L. Griffiths. Generalization, similarity, and Bayesian inference. *BBS*, 24:629–641, 2001.

[14] S. Muggleton and L. De Raedt. Inductive logic programming: theory and methods. *Journal of Logic Programming*, 19-20:629–679, 1994.

[15] N. Chomsky. *The logical structure of linguistic theory*. University of Chicago Press, Chicago, 1975.

[16] E. L. J. Leeuwenberg. A perceptual coding language for visual and auditory patterns. *American Journal of Psychology*, 84(3):307–349, 1971.

[17] B. Edmonds. *Syntactic measures of complexity*. PhD thesis, University of Manchester, 1999.

[18] W. McCune. Mace4 reference manual and guide. Technical Report ANL/MCS-TM-264, Argonne National Laboratory, 2003.

[19] L. Gerken. Decisions, decisions: infant language learning when multiple generalizations are possible. *Cognition*, 98(3):67–74, 2006.

